# Stock Selection via Nonlinear Multi-Factor Models

**Asriel U. Levin**
BZW Barclays Global Investors
Advanced Strategies and Research Group
45 Fremont Street
San Francisco CA 94105
email: asriel.levin@bglobal.com

## Abstract

This paper discusses the use of multilayer feedforward neural networks for predicting a stock's excess return based on its exposure to various technical and fundamental factors. To demonstrate the effectiveness of the approach a hedged portfolio which consists of equally capitalized long and short positions is constructed and its historical returns are benchmarked against T-bill returns and the S&P500 index.

## 1  Introduction

Traditional investment approaches (Elton and Gruber, 1991) assume that the return of a security can be described by a multifactor linear model:

$$R_i = a_i + u_{i1}F_1 + u_{i2}F_2 + \ldots + u_{iL}F_L + e_i \tag{1}$$

where $R_i$ denotes the return on security $i$, $F_l$ are a set of factor values and $u_{il}$ are security $i$ exposure to factor $l$, $a_i$ is an intercept term (which under the CAPM framework is assumed to be equal to the risk free rate of return (Sharpe, 1984)) and $e_i$ is a random term with mean zero which is assumed to be uncorrelated across securities.

The factors may consist of any set of variables deemed to have explanatory power for security returns. These could be aspects of macroeconomics, fundamental security analysis, technical attributes or a combination of the above. The value of a factor is the expected excess return above risk free rate of a security with unit exposure to the factor and zero exposure to all other factors. The choice of factors can be viewed as a proxy for the "state of the world" and their selection defines a metric imposed on the universe of securities: Once the factors are set, the model assumption is that,

on average, two securities with similar factor loadings ($u_{il}$) will behave in a similar manner.

The factor model (1) was not originally developed as a predictive model, but rather as an explanatory model, with the returns $R_i$ and the factor values $F_l$ assumed to be contemporaneous. To utilize (1) in a predictive manner, each factor value must be replaced by an estimate, resulting in the model

$$R_i = a_i + u_{i1}\hat{F}_1 + u_{i2}\hat{F}_2 + \ldots + u_{iL}\hat{F}_L + e_i \qquad (2)$$

where $R_i$ is a security's future return and $\hat{F}_l$ is an estimate of the future value of factor $l$, based on currently available information. The estimation of $\hat{F}_l$ can be approached with varying degree of sophistication ranging from a simple use of the historical mean to estimate the factor value (setting $\hat{F}_l(t) = \bar{F}_l$), to more elaborate approaches attempting to construct a time series model for predicting the factor values.

Factor models of the form (2) can be employed both to control risk and to enhance return. In the first case, by capturing the major sources of correlation among security returns, one can construct a well balanced portfolio which diversifies specific risk away. For the latter, if one is able to predict the likely future value of a factor, higher return can be achieved by constructing a portfolio that tilts toward "good" factors and away from "bad" ones.

While linear factor models have proven to be very useful tools for portfolio analysis and investment management, the assumption of linear relationship between factor values and expected return is quite restrictive. Specifically, the use of linear models assumes that each factor affects the return independently and hence, they ignore the possible interaction between different factors. Furthermore, with a linear model, the expected return of a security can grow without bound as its exposure to a factor increases. To overcome these shortcomings of linear models, one would have to consider more general models that allow for nonlinear relationship among factor values, security exposures and expected returns.

Generalizing (2), while maintaining the basic premise that the state of the world can be described by a vector of factor values and that the expected return of a security is determined through its coordinates in this factor world, leads to the nonlinear model:

$$R_i = \tilde{f}(u_{i1}, u_{i2}, \ldots, u_{iL}, \hat{F}_1, \hat{F}_2, \ldots, \hat{F}_L) + e_i \qquad (3)$$

where $\tilde{f}(\cdot)$ is a nonlinear function and $e_i$ is the noise unexplained by the model, or "security specific risk".

The prediction task for the nonlinear model (3) is substantially more complex than in the linear case since it requires both the estimation of future factor values as well as a determination of the unknown function $\tilde{f}$. The task can be somewhat simplified if factor estimates are replaced with their historical means:

$$
\begin{aligned}
R_i &= \tilde{f}(u_{i1}, u_{i2}, \ldots, u_{iL}, \bar{F}_1, \bar{F}_2, \ldots, \bar{F}_L) + e_i \\
&\triangleq f(u_{i1}, u_{i2}, \ldots, u_{iL}) + e_i
\end{aligned}
\qquad (4)
$$

where now $u_{il}$ are the security's factor exposure at the beginning of the period over which we wish to predict.

To estimate the unknown function $f(\cdot)$, a family of models needs to be selected, from which a model is to be identified. In the following we propose modeling the relationship between factor exposures and future returns using the class of multilayer feedforward neural networks (Hertz et al., 1991). Their universal approximation

capabilities (Cybenko, 1989; Hornik et al., 1989), as well as the existence of an effective parameter tuning method (the backpropagation algorithm (Rumelhart et al., 1986)) makes this family of models a powerful tool for the identification of nonlinear mappings and hence a natural choice for modeling (4).

## 2    The stock selection problem

Our objective in this paper is to test the ability of neural network based models of the form (4) to differentiate between attractive and unattractive stocks. Rather than trying to predict the total return of a security, the objective is to predict its performance relative to the market, hence eliminating the need to predict market directions and movements.

The data set consists of monthly historical records (1989 through 1995) for the largest 1200-1300 US companies as defined by the BARRA HiCap universe. Each data record ($\approx$1300 per month) consists of an input vector composed of a security's factor exposures recorded at the beginning of the month and the corresponding output is the security's return over the month. The factors used to build the model include Earning/Price, Book/Price, past price performance, consensus of analyst sentiments etc, which have been suggested in the financial literature as having explanatory power for security returns (e.g. (Fama and French, 1992)). To minimize risk, exposure to other unwarranted factors is controlled using a quadratic optimizer.

## 3    Model construction and testing

Potentially, changes in a price of a security are a function of a very large number of forces and events, of which only a small subset can be included in the factor model (4). All other sources of return play the role of noise whose magnitude is probably much larger than any signal that can be explained by the factor exposures. When this information is used to train a neural network, the network attempts to replicate the examples it sees and hence much of what it tries to learn will be the particular realizations of noise that appeared in the training set.

To minimize this effect, both a validation set and regularization are used in the training. The validation set is used to monitor the performance of the model with data on which it has not been trained on. By stopping the learning process when validation set error starts to increase, the learning of noise is minimized. Regularization further limits the complexity of the function realized by the network and, through the reduction of model variance, improves generalization (Levin et al., 1994).

The stock selection model is built using a rolling train/test window. First, $M$ "two layer" feedforward networks are built for each month of data (result is rather insensitive to the particular choice of $M$). Each network is trained using stochastic gradient descent with one quarter of the monthly data (randomly selected) used as a validation set. Regularization is done using principal component pruning (Levin et al., 1994). Once training is completed, the models constructed over $N$ consecutive month of data (again, result is insensitive to particular choice of $N$) are combined (thus increasing the robustness of the model (Breiman, 1994)) to predict the returns in the following month. Thus the predicted (out of sample) return of stock $i$ in month $k$ is given by

$$\hat{R}_i(k) = \frac{1}{N*M} \sum_{j=1}^{N*M} NN_{k-j}(u_{i1}^k, u_{i2}^k, \ldots, u_{iL}^k) \tag{5}$$

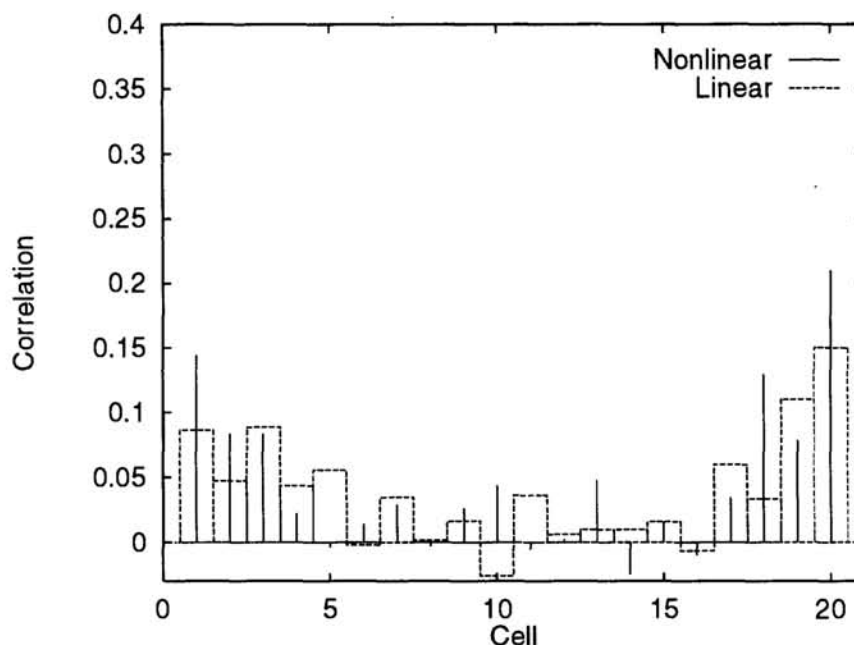

Figure 1: Average correlation between predicted alphas and realized returns for linear and nonlinear models

where $\hat{R}_i(k)$ is stock's $i$ predicted return, $NN_{k-j}(\cdot)$ denoted the neural network model built in month $k - j$ and $u_{il}^k$ are stock's $i$ factor exposures as measured at the beginning of month $k$.

## 4    Benchmarking to linear

As a first step in evaluating the added value of the nonlinear model, its performance was benchmarked against a generalized least squares linear model. Each model was run over three universes: all securities in the HiCap universe, the extreme 200 stocks (top 100, bottom 100 as defined by each model), and the extreme 100 stocks. As a comparative performance measure we use the Sharpe ratio (Elton and Gruber, 1991). As shown in Table 4, while the performance of the two models is quite comparable over the whole universe of stocks, the neural network based model performs better at the extremes, resulting in a substantially larger Sharpe ratio (and of course, when constructing a portfolio, it is the extreme alphas that have the most impact on performance).

| Portfolio\Model | Linear | Nonlinear |
|---|---|---|
| All HiCap | 6.43 | 6.92 |
| 100 long/100 short | 4.07 | 5.49 |
| 50 long/50 short | 3.07 | 4.23 |

Table 1: Ex ante Sharpe ratios: Neural network vs. linear

While the numbers in the above table look quite impressive, it should be emphasised that they do not represent returns of a practical strategy: turnover is huge and the figures do not take transaction costs into account. The main purpose of the table

is to compare the information that can be captured by the different models and specifically to show the added value of the neural network at the extremes. A practical implementation scheme and the associated performance will be discussed in the next section.

Finally, some insight as to the reason for the improved performance can be gained by looking at the correlation between model predictions and realized returns for different values of model predictions (commonly referred to as *alphas*). For that, the alpha range was divided to 20 cells, 5% of observations in each and correlations were calculated separately for each cell. As is shown in figure 1, while both neural network and linear model seem to have more predictive power at the extremes, the network's correlations are substantially larger for both positive and negative alphas.

## 5  Portfolio construction

Given the superior predictive ability of the nonlinear model at the extremes, a natural way of translating its predictions into an investment strategy is through the use of a long/short construct which fully captures the model information on both the positive as well as the negative side.

The long/short portfolio (Jacobs and Levy, 1993) is constructed by allocating equal capital to long and short positions. By monitoring and controlling the risk characteristics on both sides, one is able to construct a portfolio that has zero correlation with the market ($\beta = 0$) - a "market neutral" portfolio. By construction, the return of a market neutral portfolio is insensitive to the market up or down swings and its only source of return is the performance spread between the long and short positions, which in turn is a direct function of the model (5) discernment ability.

Specifically, the translation of the model predictions into a realistically implementable strategy is done using a quadratic optimizer. Using the model predicted returns and incorporating volatility information about the various stocks, the optimizer is utilized to construct a portfolio with the following characteristics:

- Market neutral (equal long and short capitalization).
- Total number of assets in the portfolio $<= 200$.
- Average (one sided) monthly turnover $\approx 15\%$.
- Annual active risk $\approx 5\%$.

In the following, all results are test set results (out of sample), net of estimated transaction costs (assumed to be 1.5% round trip). The standard benchmark for a market neutral portfolio is the return on 3 month T-bill and as can be seen in Table 2, over the test period the market neutral portfolio has consistently and decisively outperformed its benchmark. Furthermore, the results reported for 1995 were recorded in real-time (simulated paper portfolio).

An interesting feature of the long/short construct is its ease of transportability (Jacobs and Levy, 1993). Thus, while the base construction is insensitive to market movement, if one wishes, full exposure to a desired market can be achieved through the use of futures or swaps (Hull, 1993). As an example, by adding a permanent S&P500 futures overlay in an amount equal to the invested capital, one is fully exposed to the equity market at all time, and returns are the sum of the long/short performance spread and the profits or losses resulting from the market price movements. This form of a long/short strategy is referred to as an "equitized" strategy and the appropriate benchmark will be overlayed index. The relative performance

| Statistics | T-Bill | Neutral | S&P500 | Equitized |
|---|---|---|---|---|
| Total Return(%) | 27.8 | 131.5 | 102.0 | 264.5 |
| Annual total(Yr%) | 4.6 | 16.8 | 10.4 | 27.0 |
| Active Return(%) | - | 103.7 | - | 162.5 |
| Annual active(Yr%) | - | 12.2 | - | 16.6 |
| Active risk(Yr%) | - | 4.8 | - | 4.8 |
| Max draw down(%) | - | 3.2 | 13.9 | 10.0 |
| Turnover(Yr%) | - | 198.4 | - | 198.4 |

Table 2: Comparative summary of ex ante portfolio performance (net of transaction costs) 8/90 - 12/95

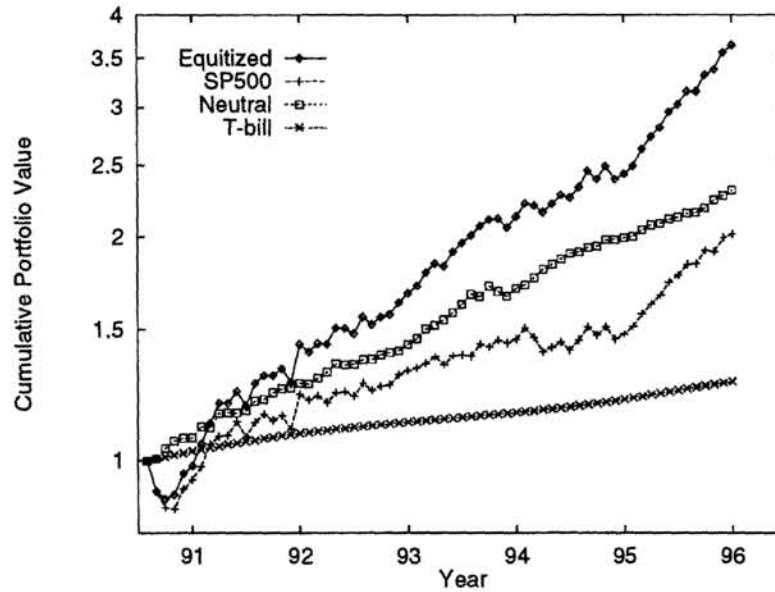

Figure 2: Cumulative portfolio value 8/90 - 12/95 (net of estimated transaction costs)

of the equitized strategy with an S&P500 futures overlay is presented in Table 2. Summary of the accumulated returns over the test period for the market neutral and equitized portfolios compared to T-bill and S&P500 are given in Figure 2.

Finally, even though the performance of the model is quite good, it is very difficult to convince an investor to put his money on a "black box". A rather simple way to overcome this problem of neural networks is to utilize a CART tree (Breiman et al., 1984) to explain the model's structure. While the performance of the tree on the raw data in substantially inferior to the network's, it can serve as a very effective tool for analyzing and interpreting the information that is driving the model.

## 6   Conclusion

We presented a methodology by which neural network based models can be used for security selection and portfolio construction. In spite of the very low signal to noise ratio of the raw data, the model was able to extract meaningful relationship

between factor exposures and expected returns. When utilized to construct hedged portfolios, these predictions achieved persistent returns with very favorable risk characteristics.

The model is currently being tested in real time and given its continued consistent performance, is expected to go live soon.

# References

Anderson, J. and Rosenfeld, E., editors (1988). *Neurocomputing: Foundations of Research*. MIT Press, Cambridge.

Breiman, L. (1994). Bagging predictors. Technical Report 416, Department of Statistics, UCB, Berkeley, CA.

Breiman, L., Friedman, J., Olshen, R., and Stone, C. (1984). *Classification and Regression Trees*. Chapman & Hall.

Cybenko, G. (1989). Approximation by superpositions of a sigmoidal function. *Mathematics of Control, Signals, and Systems*, 2:303–314.

Elton, E. and Gruber, M. (1991). *Modern Portfolio Theory and Investment Analysis*. John Wiley.

Fama, E. and French, K. (1992). The cross section of expected stock returns. *Journal of Finance*, 47:427–465.

Hertz, J., Krogh, A., and Palmer, R. (1991). *Introduction to the theory of neural computation*, volume 1 of *Santa Fe Institute studies in the sciences ofcomplexity*. Addison Wesley Pub. Co.

Hornik, K., Stinchcombe, M., and White, H. (1989). Multilayer feedforward networks are universal approximators. *Neural Networks*, 2:359–366.

Hull, J. (1993). *Options, Futures and Other Derivative Securities*. Prentice-Hall.

Jacobs, B. and Levy, K. (1993). Long/short equity investing. *Journal of Portfolio Management*, pages 52–63.

Levin, A. U., Leen, T. K., and Moody, J. E. (1994). Fast pruning using principal components. In Cowan, J. D., Tesauro, G., and Alspector, J., editors, *Advances in Neural Information Processing Systems*, volume 6. Morgan Kaufmann. to apear.

Rumelhart, D., Hinton, G., and Williams, R. (1986). Learning representations by back-propagating errors. *Nature*, 323:533–536. Reprinted in (Anderson and Rosenfeld, 1988).

Sharpe, W. (1984). Factor models, CAPMs and the APT. *Journal of Portfolio Management*, pages 21–25.
